# Trans-dimensional MCMC for Bayesian Policy Learning

**Matt Hoffman**
Dept. of Computer Science
University of British Columbia
hoffmanm@cs.ubc.ca

**Arnaud Doucet**
Depts. of Statistics and Computer Science
University of British Columbia
arnaud@cs.ubc.ca

**Nando de Freitas**
Dept. of Computer Science
University of British Columbia
nando@cs.ubc.ca

**Ajay Jasra**
Dept. of Mathematics
Imperial College London
ajay.jasra@imperial.ac.uk

## Abstract

A recently proposed formulation of the stochastic planning and control problem as one of parameter estimation for suitable artificial statistical models has led to the adoption of inference algorithms for this notoriously hard problem. At the algorithmic level, the focus has been on developing Expectation-Maximization (EM) algorithms. In this paper, we begin by making the crucial observation that the stochastic control problem can be reinterpreted as one of trans-dimensional inference. With this new interpretation, we are able to propose a novel reversible jump Markov chain Monte Carlo (MCMC) algorithm that is more efficient than its EM counterparts. Moreover, it enables us to implement full Bayesian policy search, without the need for gradients and with one single Markov chain. The new approach involves sampling directly from a distribution that is proportional to the reward and, consequently, performs better than classic simulations methods in situations where the reward is a rare event.

## 1 Introduction

Continuous state-space Markov Decision Processes (MDPs) are notoriously difficult to solve. Except for a few rare cases, including linear Gaussian models with quadratic cost, there is no closed-form solution and approximations are required [4]. A large number of methods have been proposed in the literature relying on value function approximation and policy search; including [3, 10, 14, 16, 18]. In this paper, we follow the policy learning approach because of its promise and remarkable success in complex domains; see for example [13, 15]. Our work is strongly motivated by a recent formulation of stochastic planning and control problems as inference problems. This line of work appears to have been initiated in [5], where the authors used EM as an alternative to standard stochastic gradient algorithms to maximize an expected cost. In [2], a planning problem under uncertainty was solved using a Viterbi algorithm. This was later extended in [21]. In these works, the number of time steps to reach the goal was fixed and the plans were not optimal in expected reward. An important step toward surmounting these limitations was taken in [20, 19]. In these works, the standard discounted reward control problem was expressed in terms of an infinite mixture of MDPs. To make the problem tractable, the authors proposed to truncate the infinite horizon time.

Here, we make the observation that, in this probabilistic interpretation of stochastic control, the objective function can be written as the expectation of a positive function with respect to a trans-dimensional probability distribution, *i.e.* a probability distribution defined on a union of subspaces

of different dimensions. By reinterpreting this function as a (artificial) marginal likelihood, it is easy to see that it can also be maximized using an EM-type algorithm in the spirit of [5]. However, the observation that we are dealing with a trans-dimensional distribution enables us to go beyond EM. We believe it creates many opportunities for exploiting a large body of sophisticated inference algorithms in the decision-making context.

In this paper, we propose a full Bayesian policy search alternative to the EM algorithm. In this approach, we set a prior distribution on the set of policy parameters and derive an artificial posterior distribution which is proportional to the prior times the expected reward. In the simpler context of myopic Bayesian experimental design, a similar method was developed in [11] and applied successfully to high-dimensional problems [12]. Our method can be interpreted as a trans-dimensional extension of [11]. We sample from the resulting artificial posterior distribution using a single trans-dimensional MCMC algorithm, which only involves a simple modification of the MCMC algorithm developed to implement the EM.

Although the Bayesian policy search approach can benefit from gradient information, it does not require gradients. Moreover, since the target is proportional to the expected reward, the simulation is guided to areas of high reward automatically. In the fixed policy case, the value function is often computed using importance sampling. In this context, our algorithm could be reinterpreted as an MCMC algorithm sampling from the optimal importance distribution.

## 2 Model formulation

We consider the following class of discrete-time Markov decision processes (MDPs):

$$
\begin{aligned}
X_1 &\sim \mu(\cdot) \\
X_n | (X_{n-1} = x, A_{n-1} = a) &\sim f_a(\cdot | x) \\
R_n | (X_n = x, A_n = a) &\sim g_a(\cdot | x) \\
A_n | (X_n = x, \theta) &\sim \pi_\theta(\cdot | x),
\end{aligned}
\tag{1}
$$

where $n = 1, 2, \ldots$ is a discrete-time index, $\mu(\cdot)$ is the initial state distribution, $\{X_n\}$ is the $\mathcal{X}$−valued state process, $\{A_n\}$ is the $\mathcal{A}$−valued action process, $\{R_n\}$ is a positive real-valued reward process, $f_a$ denotes the transition density, $g_a$ the reward density and $\pi_\theta$ is a randomized policy. If we have a deterministic policy then $\pi_\theta(a | x) = \delta_{\varphi_\theta(x)}(a)$. In this case, the transition model $f_a(\cdot | x)$ assumes the parametrization $f_\theta(\cdot | x)$. The reward model could also be parameterized as $g_\theta(\cdot | x)$. It should be noted that for this work we will be working within a model-based framework and as a result will require knowledge of the transition model (although it could be learned).

We are here interested in maximizing with respect to the parameters of the policy $\theta$ the expected future reward

$$
V_\mu^\pi(\theta) = \mathbb{E}\left[\sum_{n=1}^{\infty} \gamma^{n-1} R_n\right],
$$

where $0 < \gamma < 1$ is a discount factor and the expectation is with respect to the probabilistic model defined in (1). As shown in [20], it is possible to re-write this objective of optimizing an infinite horizon discounted reward MDP (where the reward happens at each step) as one of optimizing an infinite mixture of finite horizon MDPs (where the reward only happens at the last time step).

In particular, we note that by introducing the trans-dimensional probability distribution on $\biguplus \{k\} \times \mathcal{X}^k \times \mathcal{A}^k \times \mathbb{R}^+$ given by

$$
p_\theta(k, x_{1:k}, a_{1:k}, r_k) = (1 - \gamma)\gamma^{k-1}\mu(x_1) g_{a_k}(r_k | x_k) \prod_{n=2}^{k} f_{a_{n-1}}(x_n | x_{n-1}) \prod_{n=1}^{k} \pi_\theta(a_n | x_n),
\tag{2}
$$

we can easily rewrite $V_\mu^\pi(\theta)$ as an infinite mixture model of finite horizon MDPs, with the reward happening at the last horizon step; namely at $k$. Specifically we have:

$$
V_\mu^\pi(\theta) = (1 - \gamma)^{-1} \mathbb{E}_{p_\theta}[R_K] = (1 - \gamma)^{-1} \sum_{k=1}^{\infty} \int r_k p_\theta(k, x_{1:k}, a_{1:k}, r_k)\, dx_{1:k} da_{1:k} dr_k
\tag{3}
$$

for a randomized policy. Similarly, for a deterministic policy, the representation (3) also holds for the trans-dimensional probability distribution defined on $\biguplus \{k\} \times \mathcal{X}^k \times \mathbb{R}^+$ given by

$$p_\theta (k, x_{1:k}, r_k) = (1 - \gamma) \gamma^{k-1} \mu (x_1) g_\theta (r_k | x_k) \prod_{n=2}^{k} f_\theta (x_n | x_{n-1}). \tag{4}$$

The representation (3) was also used in [6] to compute the value function through MCMC for a fixed $\theta$. In [20], this representation is exploited to maximize $V_\mu^\pi (\theta)$ using the EM algorithm which, applied to this problem, proceeds as follows at iteration $i$

$$\theta_i = \underset{\theta \in \Theta}{\arg\max} \, Q (\theta_{i-1}, \theta)$$

where

$$Q (\theta_{i-1}, \theta) = \mathbb{E}_{\widetilde{p}_{\theta_{i-1}}} \left[ \log \left( R_K . p_\theta (K, X_{1:K}, A_{1:K}, R_K) \right) \right],$$

$$\widetilde{p}_\theta (k, x_{1:k}, a_{1:k}, r_k) = \frac{r_k p_\theta (k, x_{1:k}, a_{1:k}, r_k)}{\mathbb{E}_{p_\theta} [R_K]}.$$

Unlike [20], we are interested in problems with potentially nonlinear and non-Gaussian properties. In these situations, the $Q$ function cannot be calculated exactly. The standard Monte Carlo EM approach consists of sampling from $\widetilde{p}_\theta (k, x_{1:k}, a_{1:k}, r_k)$ using MCMC to obtain a Monte Carlo estimate of the $Q$ function. *As $\widetilde{p}_\theta (k, x_{1:k}, a_{1:k}, r_k)$ is proportional to the reward, the samples will consequently be drawn in regions of high reward.* This is a particularly interesting feature in situations where the reward function is concentrated in a region of low probability mass under $p_\theta (k, x_{1:k}, r_k)$, which is often the case in high-dimensional control settings. Note that if we wanted to estimate $V_\mu^\pi (\theta)$ using importance sampling, then the distribution $\widetilde{p}_\theta (k, x_{1:k}, a_{1:k}, r_k)$ corresponds to the optimal zero-variance importance distribution.

Alternatively, instead of sampling from $\widetilde{p}_\theta (k, x_{1:k}, a_{1:k}, r_k)$ using MCMC, we could proceed as in [20] to derive forward-backward algorithms to implement the E-step which can be implemented here using Sequential Monte Carlo (SMC) techniques. We have in fact done this using the smoothing algorithms proposed in [9]. However, we will focus the discussion on a different MCMC approach based on trans-dimensional simulation. As shown in the experiments, the latter does considerably better.

Finally, we remark that for a deterministic policy, we can introduce the trans-dimensional distribution:

$$\widetilde{p}_\theta (k, x_{1:k}, r_k) = \frac{r_k p_\theta (k, x_{1:k}, r_k)}{\mathbb{E}_{p_\theta} [R_K]}.$$

In addition, and for ease of presentation only, we focus the discussion on deterministic policies and reward functions $g_\theta (r_n | x_n) = \delta_{r(x_n)} (r_n)$; the extension of our algorithms to the randomized case is straightforward.

## 3 Bayesian policy exploration

The EM algorithm is particularly sensitive to initialization and might get trapped in a severe local maximum of $V_\mu^\pi (\theta)$. Moreover, in the general state-space setting that we are considering, the particle smoothers in the E-step can be very expensive computationally.

To address these concerns, we propose an alternative full Bayesian approach. In the simpler context of experimental design, this approach was successfully developed in [11], [12]. The idea consists of introducing a vague prior distribution $p (\theta)$ on the parameters of the policy $\theta$. We then define the new artificial probability distribution defined on $\Theta \times \biguplus \{k\} \times \mathcal{X}^k$ by

$$\overline{p} (\theta, k, x_{1:k}) \propto r (x_k) p_\theta (k, x_{1:k}) p (\theta).$$

By construction, this target distribution admits the following marginal in $\theta$

$$\overline{p} (\theta) \propto V_\mu^\pi (\theta) p (\theta)$$

and we can select an improper prior distribution $p (\theta) \propto 1$ if $\int_\Theta V_\mu^\pi (\theta) \, d\theta < \infty$.

If we could sample from $\overline{p}(\theta)$, then the generated samples $\{\theta^{(i)}\}$ would concentrate themselves in regions where $V_\mu^\pi(\theta)$ is large. We cannot sample from $\overline{p}(\theta)$ directly but we can developed a trans-dimensional MCMC algorithm which will generate asymptotically samples from $\overline{p}(\theta, k, x_{1:k})$, hence samples from $\overline{p}(\theta)$.

Our algorithm proceeds as follows. Assume the current state of the Markov chain targeting $\overline{p}(\theta, k, x_{1:k})$ is $(\theta, k, x_{1:k})$. We propose first to update the components $(k, x_{1:k})$ conditional upon $\theta$ using a combination of birth, death and update moves using the reversible jump MCMC algorithm [7, 8, 17]. Then we propose to update $\theta$ conditional upon the current value of $(k, x_{1:k})$. This can be achieved using a simple Metropolis-Hastings algorithm or a more sophisticated dynamic Monte Carlo schemes. For example, if gradient information is available, one could adopt Langevin diffusions and the hybrid Monte Carlo algorithm [1]. The overall algorithm is depicted in Figure 1. The details of the reversible jump algorithm are presented in the following section.

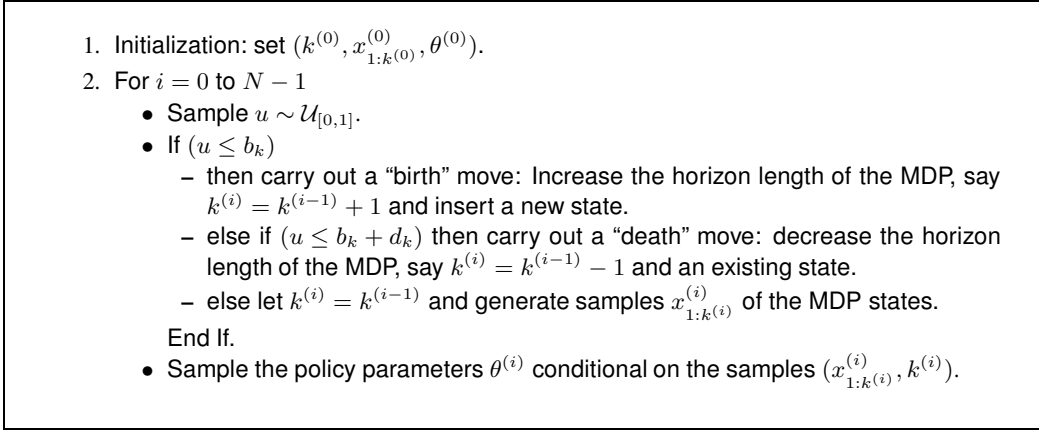

1. Initialization: set $(k^{(0)}, x_{1:k^{(0)}}^{(0)}, \theta^{(0)})$.
2. For $i = 0$ to $N - 1$
   - Sample $u \sim \mathcal{U}_{[0,1]}$.
   - If $(u \leq b_k)$
     - then carry out a "birth" move: Increase the horizon length of the MDP, say $k^{(i)} = k^{(i-1)} + 1$ and insert a new state.
     - else if $(u \leq b_k + d_k)$ then carry out a "death" move: decrease the horizon length of the MDP, say $k^{(i)} = k^{(i-1)} - 1$ and an existing state.
     - else let $k^{(i)} = k^{(i-1)}$ and generate samples $x_{1:k^{(i)}}^{(i)}$ of the MDP states.

     End If.
   - Sample the policy parameters $\theta^{(i)}$ conditional on the samples $(x_{1:k^{(i)}}^{(i)}, k^{(i)})$.

Figure 1: Generic reversible jump MCMC for Bayesian policy learning.

We note that for a given $\theta$ the samples of the states and horizon generated by this Markov chain will also be distributed (asymptotically) according to the trans-dimensional distribution $\widetilde{p}_\theta(k, x_{1:k})$. Hence, they can be easily adapted to generate a Monte Carlo estimate of $Q(\theta_{i-1}, \theta)$. This allows us to side-step the need for expensive smoothing algorithms in the E-step. The trans-dimensional simulation approach has the advantage that the samples will concentrate themselves automatically in regions where $\widetilde{p}_\theta(k)$ has high probability masses. Moreover, unlike in the EM framework, it is no longer necessary to truncate the time domain.

## 4 Trans-Dimensional Markov chain Monte Carlo

We present a simple reversible jump method composed of two reversible moves (birth and death) and several update moves. Assume the current state of the Markov chain targeting $\widetilde{p}_\theta(k, x_{1:k})$ is $(k, x_{1:k})$. With probability[1] $b_k$, we propose a birth move; that is we sample a location uniformly in the interval $\{1, ..., k+1\}$, i.e. $J \sim \mathcal{U}\{1, ..., k+1\}$, and propose the candidate $(k+1, x_{1:j-1}, x^*, x_{j:k})$ where $X^* \sim q_\theta(\cdot | x_{j-1:j})$. This candidate is accepted with probability $A_{birth} = \min\{1, \alpha_{birth}\}$ where we have for $j \in \{2, ..., k-1\}$

$$\alpha_{birth} = \frac{\widetilde{p}_\theta(k+1, x_{1:j-1}, x^*, x_{j:k}) d_{k+1}}{\widetilde{p}_\theta(k, x_{1:k}) b_k q_\theta(x^* | x_{j-1:j})}$$

$$= \frac{\gamma f_\theta(x^* | x_{j-1}) f_\theta(x_j | x^*) d_{k+1}}{f_\theta(x_j | x_{j-1}) b_k q_\theta(x^* | x_{j-1:j})},$$

for $j = 1$

$$\alpha_{birth} = \frac{\gamma \mu(x^*) f_\theta(x_1 | x^*) d_{k+1}}{\mu(x_1) b_k q_\theta(x^* | x_1)},$$

[1]In practice we can set the birth and death probabilities such that $b_k = d_k = u_k = 1/3$.

and $j = k + 1$

$$\alpha_{birth} = \frac{\gamma r\left(x^*\right) f_\theta\left(\left. x^* \right| x_k\right) d_{k+1}}{r\left(x_k\right) b_k q_\theta\left(\left. x^* \right| x_k\right)}.$$

With probability $d_k$, we propose a death move; that is $J \sim \mathcal{U}\{1, ..., k\}$ and we propose the candidate $(k - 1, x_{1:j-1}, x_{j+1:k})$ which is accepted with probability $A_{death} = \min\{1, \alpha_{death}\}$ where for $j \in \{2, ..., k-1\}$

$$\begin{aligned}\alpha_{death} &= \frac{\widetilde{p}_\theta\left(k - 1, x_{1:j-1}, x_{j+1:k}\right) b_{k+1} q_\theta\left(\left. x_j \right| x_{j-1:j+1}\right)}{\widetilde{p}_\theta\left(k, x_{1:k}\right) d_k} \\ &= \frac{f_\theta\left(\left. x_{j+1} \right| x_{j-1}\right) b_{k+1} q_\theta\left(\left. x_j \right| x_{j-1:j+1}\right)}{\gamma f_\theta\left(\left. x_{j+1} \right| x_j\right) f_\theta\left(\left. x_j \right| x_{j-1}\right) d_k},\end{aligned}$$

for $j = 1$

$$\alpha_{death} = \frac{\mu\left(x_2\right) q_\theta\left(\left. x_1 \right| x_2\right) b_{k+1}}{\gamma \mu\left(x_1\right) f_\theta\left(\left. x_2 \right| x_1\right) d_k},$$

and for $j = k$

$$\alpha_{death} = \frac{r\left(x_{k-1}\right) q_\theta\left(\left. x_k \right| x_{k-1}\right) b_{k+1}}{\gamma r\left(x_k\right) f_\theta\left(\left. x_k \right| x_{k-1}\right) d_k}.$$

The $\alpha_{birth}$ and $\alpha_{death}$ terms derived above can be thought of as ratios between the distribution over the newly proposed state of the chain (*i.e.* after the birth/death) and the current state. These terms must also ensure reversibility and the *dimension-matching* requirement for reversible jump MCMC. For more information see [7, 8].

Finally with probability $u_k = 1 - b_k - d_k$, we propose a standard (fixed dimensional) move where we update all or a subset of the components $x_{1:k}$ using say Metropolis-Hastings or Gibbs moves. There are many design possibilities for these moves. In general, one should block some of the variables so as to improve the mixing time of the Markov chain. If one adopts a simple one-at-a time Metropolis-Hastings scheme with proposals $q_\theta\left(\left. x^* \right| x_{j-1:j+1}\right)$ to update the $j$-th term, then the candidate is accepted with probability $A_{update} = \min\{1, \alpha_{update}\}$ where for $j \in \{2, ..., k-1\}$

$$\begin{aligned}\alpha_{update} &= \frac{\widetilde{p}_\theta\left(k, x_{1:j-1}, x^*, x_{j+1:k}\right) q_\theta\left(\left. x_j \right| x_{j-1}, x^*, x_{j+1}\right)}{\widetilde{p}_\theta\left(k, x_{1:k}\right) q_\theta\left(\left. x^* \right| x_{j-1:j+1}\right)} \\ &= \frac{f_\theta\left(\left. x^* \right| x_{j-1}\right) f_\theta\left(\left. x_{j+1} \right| x^*\right) q_\theta\left(\left. x_j \right| x_{j-1}, x^*, x_{j+1}\right)}{f_\theta\left(\left. x_j \right| x_{j-1}\right) f_\theta\left(\left. x_{j+1} \right| x_j\right) q_\theta\left(\left. x^* \right| x_{j-1:j+1}\right)},\end{aligned}$$

for $j = 1$

$$\alpha_{update} = \frac{\mu\left(x^*\right) f_\theta\left(\left. x_2 \right| x^*\right) q_\theta\left(\left. x_1 \right| x^*, x_2\right)}{\mu\left(x_1\right) f_\theta\left(\left. x_2 \right| x_1\right) q_\theta\left(\left. x^* \right| x_{1:2}\right)},$$

and for $j = k$

$$\alpha_{update} = \frac{r\left(x^*\right) f_\theta\left(\left. x^* \right| x_{k-1}\right) q_\theta\left(\left. x_k \right| x^*, x_{k-1}\right)}{r\left(x_k\right) f_\theta\left(\left. x_k \right| x_{k-1}\right) q_\theta\left(\left. x^* \right| x_{k-1:k}\right)}.$$

Under weak assumptions on the model, the Markov chain $\{K^{(i)}, X_{1:K}^{(i)}\}$ generated by this transition kernel will be irreducible and aperiodic and hence will generate asymptotically samples from the target distribution $\widetilde{p}_\theta\left(k, x_{1:k}\right)$.

We emphasize that the structure of the distributions $\widetilde{p}_\theta\left(\left. x_{1:k} \right| k\right)$ will not in many applications vary significantly with $k$ and we often have $\widetilde{p}_\theta\left(\left. x_{1:k} \right| k\right) \approx \widetilde{p}_\theta\left(\left. x_{1:k} \right| k + 1\right)$. Hence the probability of having the reversible moves accepted will be reasonable. Standard Bayesian applications of reversible jump MCMC usually do not enjoy this property and it makes it more difficult to design fast mixing algorithms. In this respect, our problem is easier.

## 5  Experiments

It should be noted from the outset that the results presented in this paper are preliminary, and serve mainly as an illustration of the Monte Carlo algorithms presented earlier. With that note aside, even these simple examples will give us some intuition about the algorithms' performance and behavior.

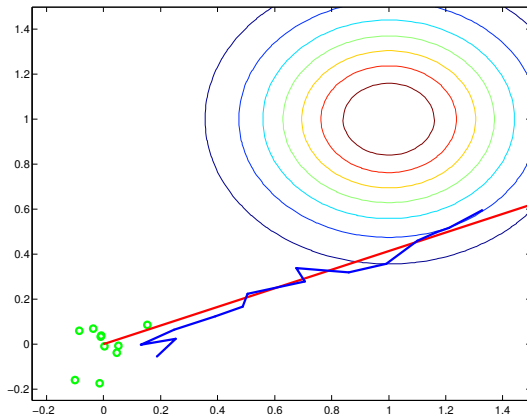

Figure 2: This figure shows an illustration of the 2d state-space described in section 5. Ten sample points are shown distributed according to $\mu$, the initial distribution, and the contour plot corresponds to the reward function $r$. The red line denotes the policy parameterized by some angle $\theta$, while a path is drawn in blue sampled from this policy.

We are also very optimistic as to the possible applications of analytic expressions for linear Gaussian models, but space has not allowed us to present simulations for this class of models here.

We will consider state- and action-spaces $\mathcal{X} = \mathcal{A} = \mathbb{R}^2$ such that each state $x \in \mathcal{X}$ is a 2d position and each action $a \in \mathcal{A}$ is a vector corresponding to a change in position. A new state at time $n$ is given by $X_n = X_{n-1} + A_{n-1} + \nu_{n-1}$ where $\nu_{n-1}$ denotes zero-mean Gaussian noise. Finally we will let $\mu$ be a normal distribution about the origin, and consider a reward (as in [20]) given by an unnormalized Gaussian about some point $m$, i.e. $r(x) = \exp(-\frac{1}{2}(x - m)^T \Sigma^{-1}(x - m))$. An illustration of this space can be seen in Figure 2 where $m = (1, 1)$.

For these experiments we chose a simple, stochastic policy parameterized by $\theta \in [0, 2\pi]$. Under this policy, an action $A_n = (w + \delta) \cdot (\cos(\theta + \omega), \sin(\theta + \omega))$ is taken where $\delta$ and $\omega$ are normally distributed random variables and $w$ is some (small) constant step-length. Intuitively, this policy corresponds to choosing a direction $\theta$ in which the agent will walk. While unrealistic from a real-world perspective, this allows us a method to easily evaluate and plot the convergence of our algorithm. For a state-space with initial distribution and reward function defined as in Figure 2 the optimal policy corresponds to $\theta = \pi/4$.

We first implemented a simple SMC-based extension of the EM algorithm described in [20], wherein a particle filter was used for the forwards/backwards filters. The plots in Figure 3 compare the SMC-based and trans-dimensional approaches performing on this synthetic example. Here the inferred value of $\theta$ is shown against CPU time, averaged over 5 runs. The first thing of note is the terrible performance of the SMC-based algorithm—in fact we had to make the reward broader and closer to the initial position in order to ensure that the algorithm converges in a reasonable amount of time. This comes as no surprise considering the $O(N^2 k_{\max}^2)$ time complexity necessary for computing the importance weights. While there do exist methods [9] for reducing this complexity to $O(N \log N k_{\max}^2)$, the discrepancy between this and the reversible jump MCMC method suggests that the MCMC approach may be more adapted to this class of problems. In the finite/discrete case it is also possible, as shown by Toussaint *et al* (2006), to reduce the $k_{\max}^2$ term to $k_{\max}$ by calculating updates only using messages from the backwards recursion. The SMC method might further be improved by better choices for the artificial distribution $\eta_n(x_n)$ in the backwards filter. In this problem we used a vague Gaussian centered on the relevant state-space. It is however possible that any added benefit from a more informative $\eta$ distribution is counterbalanced by the time required to calculate this $\eta$, for example by simulating particles forward in order to find the invariant distribution, etc.

Also shown in figure 3 is the performance of a Monte Carlo EM algorithm using reversible jump MCMC in the E-step. Both this and the fully Bayesian approach perform comparably, although the fully Bayesian approach shows less in-run variance, as well as less variance between runs. The EM algorithm was also more sensitive, and we were forced to increase the number of samples $N$ used

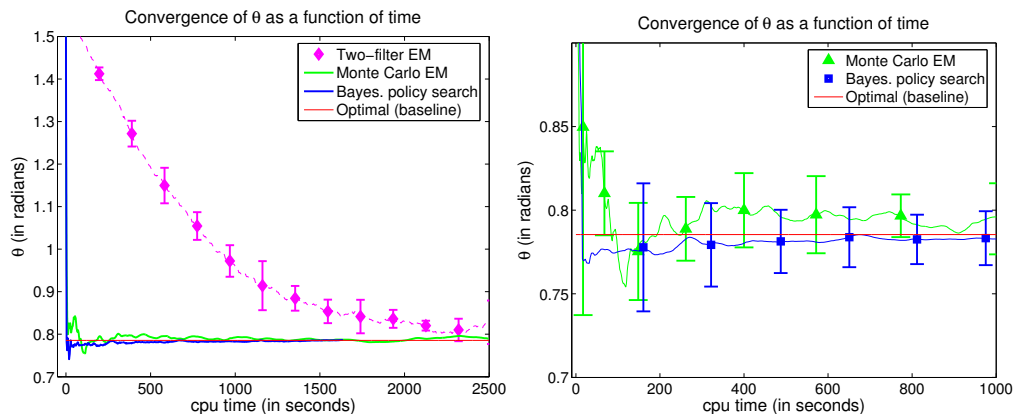

Figure 3: The left figure shows estimates for the policy parameter $\theta$ as a function of the CPU time used to calculate that value. This data is shown for the three discussed Monte Carlo algorithms as applied to a synthetic example and has been averaged over five runs; error bars are shown for the SMC-based EM algorithm. Because of the poor performance of the SMC-based algorithm it is difficult to compare the performance of the other two algorithms using only this plot. The right figure shows a smoothed and "zoomed" version of the right plot in order to show the reversible-jump EM algorithm and the fully Bayesian algorithm in more detail. In both plots a red line denotes the known optimal policy parameter of $\pi/4$.

by the E-step as the algorithm progressed, as well as controlling the learning rate with a smoothing parameter. For higher dimensional and/or larger models it is not inconceivable that this could have an adverse affect on the algorithms performance.

Finally, we also compared the proposed Bayesian policy exploration method to the PEGASUS [14] approach using a local search method. We initially tried using a policy-gradient approach, but because of the very highly-peaked rewards the gradients become very poorly scaled and would have required more tuning. As shown in Figure 4, the Bayesian strategy is more efficient in this rare event setting. As the dimension of the state-space increases, we expect this difference to become even more pronounced.

## 6 Discussion

We believe that formulating stochastic control as a trans-dimensional inference problem is fruitful. This formulation relies on minimal assumptions and allows us to apply modern inference algorithms to solve control problems. We have focused here on Monte Carlo methods and have presented— to the best of our knowledge—the first application of reversible jump MCMC to policy search. Our results, on an illustrative example, showed that this trans-dimensional MCMC algorithm is more effective that standard policy search methods and alternative Monte Carlo methods relying on particle filters. However, this methodology remains to be tested on high-dimensional problems. For such scenarios, we expect that it will be necessary to develop more efficient MCMC strategies to explore the policy space efficiently.

## References

[1] C. Andrieu, N. de Freitas, A. Doucet, and M. I. Jordan. An introduction to MCMC for machine learning. *Machine Learning*, 50:5–43, 2003.

[2] H. Attias. Planning by probabilistic inference. In *Uncertainty in Artificial Intelligence*, 2003.

[3] J. Baxter and P. L. Bartlett. Infinite-horizon policy-gradient estimation. *Journal of Artificial Intelligence Research*, 15:319–350, 2001.

[4] D. P. Bertsekas. *Dynamic Programming and Optimal Control*. Athena Scientific, 1995.

[5] P. Dayan and G. E. Hinton. Using EM for reinforcement learning. *Neural Computation*, 9:271–278, 1997.

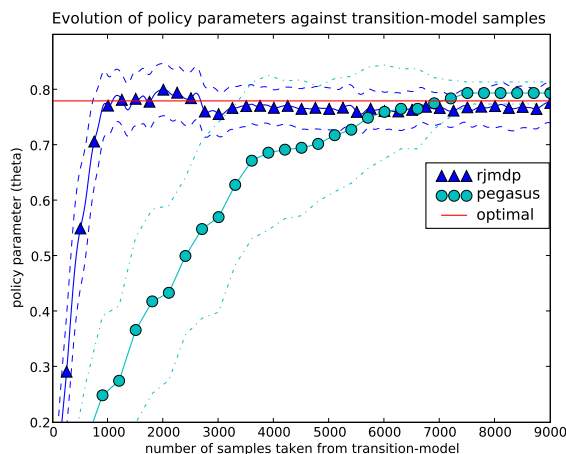

Figure 4: Convergence of PEGASUS and our Bayesian policy search algorithm when started from $\theta = 0$ and converging to the optimum of $\theta^* = \pi/4$. The plots are averaged over 10 runs. For our algorithm we plot samples taken directly from the MCMC algorithm itself: plotting the empirical average would produce an estimate whose convergence is almost immediate, but we also wanted to show the "burn-in" period. For both algorithms lines denoting one standard deviation are shown and performance is plotted against the number of samples taken from the transition model.

[6] A. Doucet and V. B. Tadic. On solving integral equations using Markov chain Monte Carlo methods. Technical Report CUED-F-INFENG 444, Cambridge University Engineering Department, 2004.

[7] P. J. Green. Reversible jump Markov chain Monte Carlo computation and Bayesian model determination. *Biometrika*, 82:711–732, 1995.

[8] P. J. Green. Trans-dimensional Markov chain Monte Carlo. In *Highly Structured Stochastic Systems*, 2003.

[9] M. Klaas, M. Briers, N. de Freitas, A. Doucet, and S. Maskell. Fast particle smoothing: If i had a million particles. In *International Conference on Machine Learning*, 2006.

[10] G. Lawrence, N. Cowan, and S. Russell. Efficient gradient estimation for motor control learning. In *Uncertainty in Artificial Intelligence*, pages 354–36, 2003.

[11] P. Müller. Simulation based optimal design. *Bayesian Statistics*, 6, 1999.

[12] P. Müller, B. Sansó, and M. De Iorio. Optimal Bayesian design by inhomogeneous Markov chain simulation. *J. American Stat. Assoc.*, 99:788–798, 2004.

[13] A. Ng, A. Coates, M. Diel, V. Ganapathi, J. Schulte, B. Tse, E. Berger, and E. Liang. Inverted autonomous helicopter flight via reinforcement learning. In *International Symposium on Experimental Robotics*, 2004.

[14] A. Y. Ng and M. I. Jordan. PEGASUS: A policy search method for large MDPs and POMDPs. In *Uncertainty in Artificial Intelligence*, 2000.

[15] J. Peters and S. Schaal. Policy gradient methods for robotics. In *IEEE International Conference on Intelligent Robotics Systems*, 2006.

[16] M. Porta, N. Vlassis, M. T. J. Spaan, and P. Poupart. Point-based value iteration for continuous POMDPs. *Journal of Machine Learning Research*, 7:2329–2367, 2006.

[17] S. Richardson and P. J. Green. On Bayesian analysis of mixtures with an unknown number of components. *Journal of the Royal Statistical Society B*, 59(4):731–792, 1997.

[18] S. Thrun. Monte Carlo POMDPs. In S. Solla, T. Leen, and K.-R. Müller, editors, *Neural Information Processing Systems*, pages 1064–1070. MIT Press, 2000.

[19] M. Toussaint, S. Harmeling, and A. Storkey. Probabilistic inference for solving (PO)MDPs. Technical Report EDI-INF-RR-0934, University of Edinburgh, School of Informatics, 2006.

[20] M. Toussaint and A. Storkey. Probabilistic inference for solving discrete and continuous state Markov decision processes. In *International Conference on Machine Learning*, 2006.

[21] D. Verma and R. P. N. Rao. Planning and acting in uncertain environments using probabilistic inference. In *IEEE/RSJ Int. Conf. on Intelligent Robots and Systems*, 2006.

